# Gradient Descent: Second-Order Momentum and Saturating Error

**Barak Pearlmutter**
Department of Psychology
P.O. Box 11A Yale Station
New Haven, CT 06520-7447
pearlmutter-barak@yale.edu

## Abstract

Batch gradient descent, $\Delta w(t) = -\eta dE/dw(t)$, converges to a minimum of quadratic form with a time constant no better than $\frac{1}{4}\lambda_{\max}/\lambda_{\min}$ where $\lambda_{\min}$ and $\lambda_{\max}$ are the minimum and maximum eigenvalues of the Hessian matrix of $E$ with respect to $w$. It was recently shown that adding a momentum term $\Delta w(t) = -\eta dE/dw(t) + \alpha \Delta w(t-1)$ improves this to $\frac{1}{4}\sqrt{\lambda_{\max}/\lambda_{\min}}$, although only in the batch case. Here we show that second-order momentum, $\Delta w(t) = -\eta dE/dw(t) + \alpha \Delta w(t-1) + \beta \Delta w(t-2)$, can lower this no further. We then regard gradient descent with momentum as a dynamic system and explore a nonquadratic error surface, showing that saturation of the error accounts for a variety of effects observed in simulations and justifies some popular heuristics.

## 1   INTRODUCTION

Gradient descent is the bread-and-butter optimization technique in neural networks. Some people build special purpose hardware to accelerate gradient descent optimization of backpropagation networks. Understanding the dynamics of gradient descent on such surfaces is therefore of great practical value.

Here we briefly review the known results in the convergence of batch gradient descent; show that second-order momentum does not give any speedup; simulate a real network and observe some effect not predicted by theory; and account for these effects by analyzing gradient descent with momentum on a saturating error surface.

## 1.1  SIMPLE GRADIENT DESCENT

First, let us review the bounds on the convergence rate of simple gradient descent without momentum to a minimum of quadratic form [11, 1]. Let $w^*$ be the minimum of $E$, the error, $H = d^2E/d\mathbf{w}^2(\mathbf{w}^*)$, and $\lambda_i$, $\mathbf{v}_i$ be the eigenvalues and eigenvectors of $H$. The weight change equation

$$\Delta\mathbf{w} = -\eta\frac{dE}{d\mathbf{w}} \tag{1}$$

(where $\Delta f(t) \equiv f(t+1) - f(t)$) is limited by

$$0 < \eta < 2/\lambda_{\max} \tag{2}$$

We can substitute $\eta = 2/\lambda_{\max}$ into the weight change equation to obtain convergence that tightly bounds any achievable in practice, getting a time constant of convergence of $-1/\log(1-2s) = (2s)^{-1} + O(1)$, or

$$E - E^* \;\succ\; \exp(-4st) \tag{3}$$

where we use $s = \lambda_{\min}/\lambda_{\max}$ for the inverse eigenvalues spread of $H$ and $\succ$ is read "asymptotically converges to zero more slowly than."

## 1.2  FIRST-ORDER MOMENTUM

Sometimes a momentum term is used, the weight update (1) being modified to incorporate a momentum term $\alpha < 1$ [5, equation 16],

$$\Delta\mathbf{w}(t) = -\eta\frac{dE}{d\mathbf{w}}(t) + \alpha\Delta\mathbf{w}(t-1). \tag{4}$$

The Momentum LMS algorithm, MLMS, has been analyzed by Shynk and Roy [6], who have shown that the momentum term can not speed convergence in the online, or stochastic gradient, case. In the batch case, which we consider here, Tuğay and Tanik [9] have shown that momentum is stable when

$$\alpha < 1 \quad \text{and} \quad 0 < \eta < 2(\alpha+1)/\lambda_{\max} \tag{5}$$

which speeds convergence to

$$E - E^* \succ \exp(-(4\sqrt{s} + O(s))\,t) \tag{6}$$

by

$$\alpha^* = \frac{2 - 4\sqrt{s(1-s)}}{(1-2s)^2} - 1 = 1 - 4\sqrt{s} + O(s), \qquad \eta^* = 2(\alpha^* + 1)/\lambda_{\max}. \tag{7}$$

## 2  SECOND-ORDER MOMENTUM

The time constant of asymptotic convergence can be changed from $O(\lambda_{\max}/\lambda_{\min})$ to $O(\sqrt{\lambda_{\max}/\lambda_{\min}})$ by going from a first-order system, (1), to a second-order system, (4). Making a physical analogy, the first-order system corresponds to a circuit with

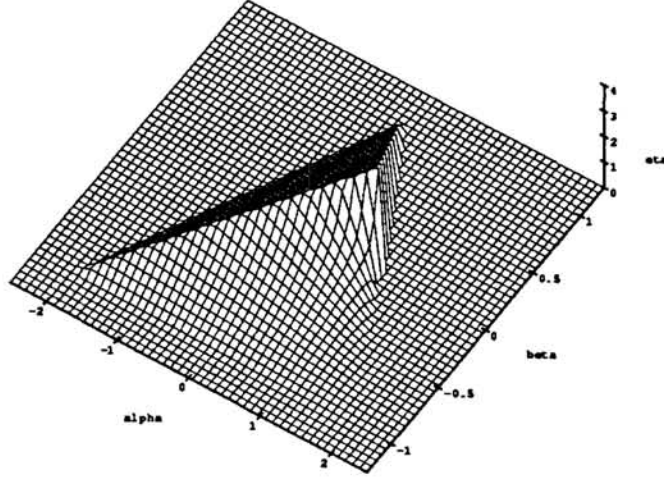

Figure 1: Second-order momentum converges if $\eta\lambda_{\max}$ is less than the value plotted as "eta," as a function of $\alpha$ and $\beta$. The region of convergence is bounded by four smooth surfaces: three planes and one hyperbola. One of the planes is parallel to the $\eta$ axis, even though the sampling of the plotting program makes it appear slightly sloped. Another is at $\eta = 0$ and thus hidden. The peak is at 4.

a resistor, and the second-order system adds a capacitor to make an RC oscillator. One might ask whether further gains can be had by going to a third-order system,

$$\Delta\mathbf{w}(t) = -\eta\frac{dE}{d\mathbf{w}} + \alpha\Delta\mathbf{w}(t-1) + \beta\Delta\mathbf{w}(t-2). \tag{8}$$

For convergence, all the eigenvalues of the matrix

$$M_i = \begin{pmatrix} 0 & 1 & 0 \\ 0 & 0 & 1 \\ -\beta & -\alpha+\beta & 1-\eta\lambda_i+\alpha \end{pmatrix}$$

in $(c_i(t-1)\ c_i(t)\ c_i(t+1))^T \approx M_i(c_i(t-2)\ c_i(t-1)\ c_i(t))^T$ must have absolute value less than or equal to 1, which occurs precisely when

$$\begin{array}{rcccl}
-1 & \leq & \beta & \leq & 1 \\
0 & \leq & \eta & \leq & 4(\beta+1)/\lambda_i \\
\eta\lambda_i/2 - (1-\beta) & \leq & \alpha & \leq & \beta\eta\lambda_i/2 + (1-\beta).
\end{array}$$

For $\beta \leq 0$ this is most restrictive for $\lambda_{\max}$, but for $\beta > 0$ $\lambda_{\min}$ also comes into play. Taking the limit as $\lambda_{\min} \to 0$, this gives convergence conditions for gradient descent with second-order momentum of

$$\begin{array}{rcl}
-1 \leq & \beta & \\
\beta - 1 \leq & \alpha & \leq 1-\beta
\end{array}$$

when $\alpha \leq 3\beta + 1$:

$$0 \leq\ \eta\ \leq \frac{2}{\lambda_{\max}}(1+\alpha-\beta) \tag{9}$$

when $\alpha \geq 3\beta + 1$:

$$0 \leq\ \eta\ \leq \frac{\beta+1}{\lambda_{\max}\beta}(\alpha+\beta-1)$$

a region shown in figure 1.

Fastest convergence for $\lambda_{\min}$ within this region lies along the ridge $\alpha = 3\beta + 1$, $\eta = 2(1 + \alpha - \beta)/\lambda_{\max}$. Unfortunately, although convergence is slightly faster than with first-order momentum, the relative advantage tends to zero as $s \to 0$, giving negligible speedup when $\lambda_{\max} \gg \lambda_{\min}$. For small $s$, the optimal settings of the parameters are

$$
\begin{aligned}
\alpha^{**} &= 1 - \frac{9}{4}\sqrt{s} + O(s) \\
\beta^{**} &= -\frac{3}{4}\sqrt{s} + O(s) \\
\eta^{**} &= 4(1 - \sqrt{s}) + O(s)
\end{aligned}
\tag{10}
$$

where $\alpha^*$ is as in (7).

## 3    SIMULATIONS

We constructed a standard three layer backpropagation network with 10 input units, 3 sigmoidal hidden units, and 10 sigmoidal output units. 15 associations between random 10 bit binary input and output vectors were constructed, and the weights were initialized to uniformly chosen random values between $-1$ and $+1$. Training was performed with a square error measure, batch weight updates, targets of 0 and 1, and a weight decay coefficient of 0.01.

To get past the initial transients, the network was run at $\eta = 0.45, \alpha = 0$ for 150 epochs, and at $\eta = 0.3, \alpha = 0.9$ for another 200 epochs. The weights were then saved, and the network run for 200 epochs for $\eta$ ranging from 0 to 0.5 and $\alpha$ ranging from 0 to 1 from that starting point.

Figure 3 shows that the region of convergence has the shape predicted by theory. Calculation of the eigenvalues of $d^2E/d\mathbf{w}^2$ confirms that the location of the boundary is correctly predicted. Figure 2 shows that momentum speeded convergence by the amount predicted by theory. Figure 3 shows that the parameter setting that give the most rapid convergence in practice are the settings predicted by theory.

However, within the region that does not converge to the minimum, there appear to be two regimes: one that is characterized by apparently chaotic fluctuations of the error, and one which slopes up gradually from the global minimum. Since this phenomenon is so atypical of a quadratic minimum in a linear system, which either converges or diverges, and this phenomenon seems important in practice, we decided to investigate a simple system to see if this behavior could be replicated and understood, which is the subject of the next section.

## 4    GRADIENT DESCENT WITH SATURATING ERROR

The analysis of the sections above may be objected to on the grounds that it assumes the minimum to have quadratic form and then performs an analysis in the neighborhood of that minimum, which is equivalent to analyzing a linear unit. Surely our nonlinear backpropagation networks are richer than that.

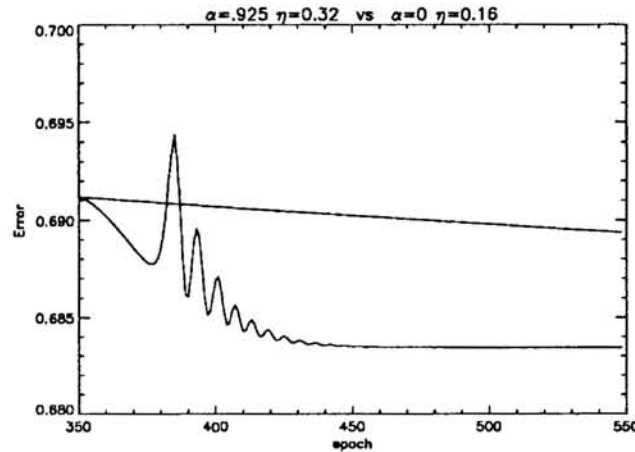

Figure 2: Error plotted as a function of time for two settings of the learning parameters, both determined empirically: the one that minimized the error the most, and the one with $\alpha = 0$ that minimized the error the most. There exists a less aggressive setting of the parameters that converges nearly as fast as the quickly converging curve but does not oscillate.

A clue that this might be the case was shown in figure 3. The region where the system converges to the minimum is of the expected shape, but rather than simply diverging outside of this region, as would a linear system, more complex phenomena are observed, in particular a sloping region.

Acting on the hypothesis that this region is caused by $\lambda_{\max}$ being maximal at the minimum, and gradually decreasing away from it (it must decrease to zero in the limit, since the hidden units saturate and the squared error is thus bounded) we decided to perform a dynamic systems analysis of the convergence of gradient descent on a one dimensional nonquadratic error surface. We chose

$$E = 1 - \frac{1}{1 + w^2} \tag{11}$$

which is shown in figure 4, as this results in a bounded $E$.

Letting

$$f(w) = w - \eta E'(w) = \frac{w(1 - 2\eta + 2w^2 + w^4)}{(1 + w^2)^2} \tag{12}$$

be our transfer function, a local analysis at the minimum gives $\lambda_{\max} = E''(0) = 2$ which limits convergence to $\eta < 1$. Since the gradient towards the minimum is always less than predicted by a second-order series at the minimum, such $\eta$ are in fact globally convergent. As $\eta$ passes 1 the fixedpoint bifurcates into the limit cycle

$$w = \pm\sqrt{\sqrt{\eta} - 1}, \tag{13}$$

which remains stable until $\eta \to 16/9 = 1.77777\ldots$, at which point the single symmetric binary limit cycle splits into two asymmetric limit cycles, each still of period two. These in turn remain stable until $\eta \to 2.0732261475-$, at which point repeated period doubling to chaos occurs. This progression is shown in figure 7.

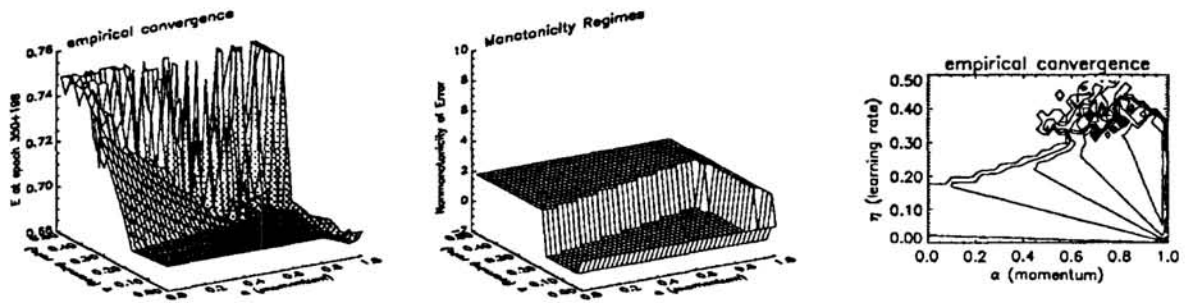

Figure 3: (Left) the error at epoch 550 as a function of the learning regime. Shading is based on the height, but most of the vertical scale is devoted to nonconvergent networks in order to show the mysterious nonconvergent sloping region. The minimum, corresponding to the most darkly shaded point, is on the plateau of convergence at the location predicted by the theory. (Center) the region in which the network is convergent, as measured by a strictly monotonically decreasing error. Learning parameter settings for which the error was strictly decreasing have a low value while those for which it was not have a high one. The lip at $\eta = 0$ has a value of 0, given where the error did not change. The rim at $\alpha = 1$ corresponds to damped oscillation caused by $\eta > 4\alpha\lambda/(1-\alpha)^2$. (Right) contour plot of the convergent plateau shows that the regions of equal error have linear boundaries in the nonoscillatory region in the center, as predicted by theory.

As usual in a bifurcation, $w$ rises sharply as $\eta$ passes 1. But recall that figure 3, with the smooth sloping region, plotted the error $E$ rather than the weights. The analogous graph here is shown in figure 6 where we see the same qualitative feature of a smooth gradual rise, which first begins to jitter as the limit cycle becomes asymmetric, and then becomes more and more jagged as the period doubles its way to chaos. From figure 7 it is clear that for higher $\eta$ the peak error of the attractor will continue to rise gently until it saturates.

Next, we add momentum to the system. This simple one dimensional system duplicates the phenomena we found earlier, as can be seen by comparing figure 3 with figure 5. We see that momentum delays the bifurcation of the fixed point attractor at the minimum by the amount predicted by (5), namely until $\eta$ approaches $1 + \alpha$. At this point the fixedpoint bifurcates into a symmetric limit cycle of period 2 at

$$w = \pm\sqrt{\sqrt{\frac{\eta}{1+\alpha}} - 1}, \tag{14}$$

a formula of which (13) is a special case. This limit cycle is stable for

$$\eta < \frac{16}{9}(1+\alpha), \tag{15}$$

but as $\eta$ reaches this limit, which happens at the same time that $w$ reaches $\pm 1/\sqrt{3}$ (the inflection point of $E$ where $E = 1/4$) the limit cycle becomes unstable. However, for $\alpha$ near 1 the cycle breaks down more quickly in practice, as it becomes haloed by more complex attractors which make it progressively less likely that a sequence of iterations will actually converge to the limit cycle in question. Both boundaries of this strip, $\eta = 1 + \alpha$ and $\eta = \frac{16}{9}(1+\alpha)$, are visible in figure 5,

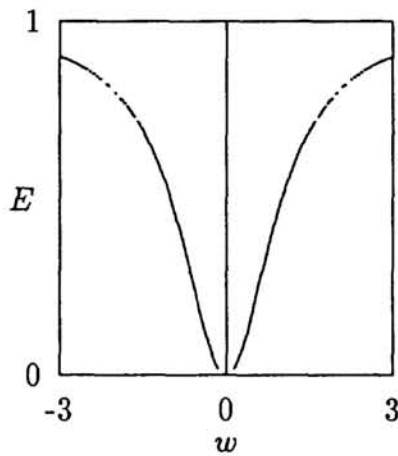

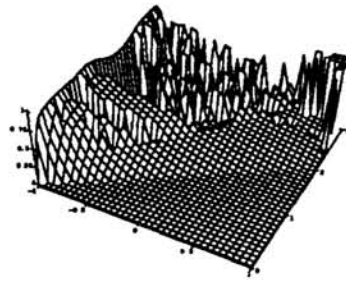

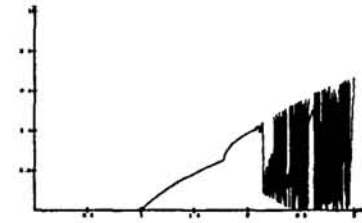

Figure 4: A one dimensional tulip-shaped nonlinear error surface $E = 1 - (1 + w^2)^{-1}$.

Figure 5: $E$ after 50 iterations from a starting point of 0.05, as a function of $\eta$ and $\alpha$.

Figure 6: $E$ as a function of $\eta$ with $\alpha = 0$. When convergent, the final value is shown; otherwise $E$ after 100 iterations from a starting point of $w = 1.0$. This a more detailed graph of a slice of figure 5 at $\alpha = 0$.

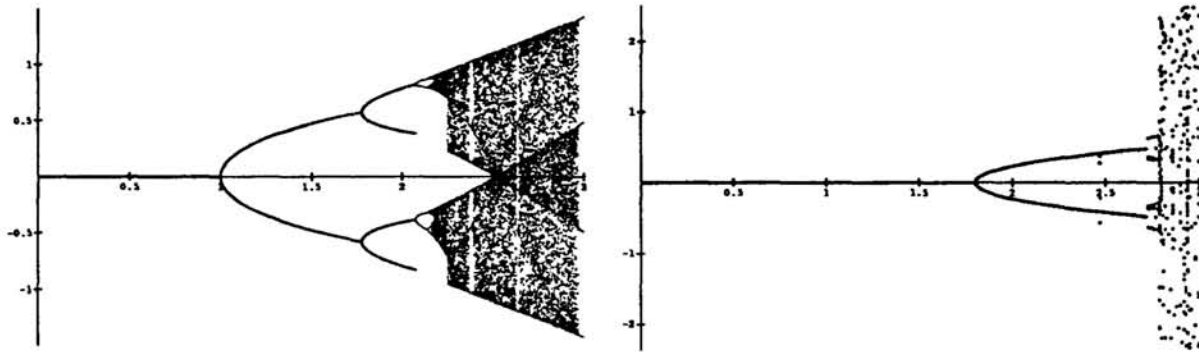

Figure 7: The attractor in $w$ as a function of $\eta$ is shown, with the progression from a single attractor at the minimum of $E$ to a limit cycle of period two, which bifurcates and then doubles to chaos. $\alpha = 0$ (left) and $\alpha = 0.8$ (right). For the numerical simulations portions of the graphs, iterations 100 through 150 from a starting point of $w = 1$ or $w = 0.05$ are shown.

particularly since in the region between them $E$ obeys

$$E = 1 - \sqrt{\frac{1 + \alpha}{\eta}} \qquad (16)$$

The bifurcation and subsequent transition to chaos with momentum is shown for $\alpha = 0.8$ in figure 7. This $\alpha$ is high enough that the limit cycle fails to be reached by the iteration procedure long before it actually becomes unstable. Note that this diagram was made with $w$ started near the minimum. If it had been started far from it, the system would usually not reach the attractor at $w = 0$ but instead enter a halo attractor. This accounts for the policy of backpropagation experts, who gradually raise momentum as the optimization proceeds.

## 5   CONCLUSIONS

The convergence bounds derived assume that the learning parameters are set optimally. Finding these optimal values in practice is beyond the scope of this paper, but some techniques for achieving nearly optimal learning rates are available [4, 10, 8, 7, 3]. Adjusting the momentum feels easier to practitioners than adjusting the learning rate, as too high a value leads to small oscillations rather than divergence, and techniques from control theory can be applied to the problem [2].

However, because error surfaces in practice saturate, techniques for adjusting the learning parameters automatically as learning proceeds can not be derived under the quadratic minimum assumption, but must take into account the bifurcation and limit cycle and the sloping region of the error, or they may mistake this regime of stable error for convergence, leading to premature termination.

## References

[1] S. Thomas Alexander. *Adaptive Signal Processing.* Springer-Verlag, 1986.

[2] H. S. Dabis and T. J. Moir. Least mean squares as a control system. *International Journal of Control*, 54(2):321–335, 1991.

[3] Yan Fang and Terrence J. Sejnowski. Faster learning for dynamic recurrent backpropagation. *Neural Computation*, 2(3):270–273, 1990.

[4] Robert A. Jacobs. Increased rates of convergence through learning rate adaptation. *Neural Networks*, 1(4):295–307, 1988.

[5] David E. Rumelhart, Geoffrey E. Hinton, and R. J. Williams. Learning internal representations by error propagation. In D. E. Rumelhart, J. L. McClelland, and the PDP research group., editors, *Parallel distributed processing: Explorations in the microstructure of cognition, Volume 1: Foundations.* MIT Press, 1986.

[6] J. J. Shynk and S. Roy. The LMS algorithm with momentum updating. In *Proceedings of the IEEE International Symposium on Circuits and Systems*, pages 2651–2654, June 6–9 1988.

[7] F. M. Silva and L. B. Almeida. Acceleration techniques for the backpropagation algorithm. In L. B. Almeida and C. J. Wellekens, editors, *Proceedings of the 1990 EURASIP Workshop on Neural Networks*. Springer-Verlag, February 1990. (Lecture Notes in Computer Science series).

[8] Tom Tollenaere. SuperSAB: Fast adaptive back propagation with good scaling properties. *Neural Networks*, 3(5):561–573, 1990.

[9] Mehmet Ali Tuğay and Yalçin Tanik. Properties of the momentum LMS algorithm. *Signal Processing*, 18(2):117–127, October 1989.

[10] T. P. Vogl, J. K. Mangis, A. K. Zigler, W. T. Zink, and D. L. Alkon. Accelerating the convergence of the back-propagation method. *Biological Cybernetics*, 59:257–263, September 1988.

[11] B. Widrow, J. M. McCool, M. G. Larimore, and C. R. Johnson Jr. Stational and nonstationary learning characteristics of the LMS adaptive filter. *Proceedings of the IEEE*, 64:1151–1162, 1979.
